# Visual Motion Computation in Analog VLSI using Pulses

Rahul Sarpeshkar, Wyeth Bair and Christof Koch
Computation and Neural Systems Program
California Institute of Technology
Pasadena, CA 91125.

## Abstract

The real time computation of motion from real images using a single chip with integrated sensors is a hard problem. We present two analog VLSI schemes that use pulse domain neuromorphic circuits to compute motion. Pulses of variable width, rather than graded potentials, represent a natural medium for evaluating temporal relationships. Both algorithms measure speed by timing a moving edge in the image. Our first model is inspired by Reichardt's algorithm in the fly and yields a non-monotonic response vs. velocity curve. We present data from a chip that implements this model. Our second algorithm yields a monotonic response vs. velocity curve and is currently being translated into silicon.

## 1 Introduction

Analog VLSI chips for the real time computation of visual motion have been the focus of much active research because of their importance as sensors for robotic applications. Correlation schemes such as those described in (Delbrück, 1993) have been found to be more robust than gradient schemes described in (Tanner and Mead, 1986), because they do not involve noise-sensitive operations like spatial-differentiation and division. A comparison of four experimental schemes may be found in (Horiuchi et al., 1992). In spite of years of work, however, there is still no motion chip that robustly computes motion under all environmental conditions.

Motion algorithms operating on higher level percepts in an image such as zero-crossings (edges) are more robust than those that operating on lower level percepts in an image such as raw image intensity values (Marr and Ullman, 1981). Our work demonstrates how, if the edges in an image are identified, it is possible to compute motion, quickly and easily, by using pulses. We compute the velocity at each point in the image. The estimation of the flow-field is of tremendous importance in computations such as time-to-contact, figure-ground-segregation and depth-from-motion. Our motion scheme is well-suited to typical indoor environments that tend to have a lot of high-contrast edges. The much harder problem of computing motion in low-contrast, high-noise outdoor environments still remains unsolved.

We present two motion algorithms. Our first algorithm is a "delay-and-correlate" scheme operating on spatial edge features and is inspired by work on fly vision (Hassenstein and Reichardt, 1956). It yields a non-monotonic response vs. velocity curve. We present data from a chip that implements it. Our second algorithm is a "facilitate-and-trigger" scheme operating on temporal edge features and yields a monotonic response vs. velocity curve. Work is under way to implement our second algorithm in analog VLSI.

## 2    The Delay-and-Correlate Scheme

Conceptually, there are two stages of computation. First, the zero-crossings in the image are computed and then the motion of these zero-crossings is detected. The zero-crossing circuitry has been described in (Bair and Koch, 1991). We concentrate on describing the motion circuitry.

A schematized version of the chip is shown in Figure 1a. Only four photoreceptors in the array are shown. The 1-D image from the array of photoreceptors is filtered with a spatial bandpass filter whose kernel is composed of a difference of two exponentials (implemented with resistive grids). The outputs of the bandpass filter feed into edge detection circuitry that output a bit indicating the presence or absence of an edge between two adjacent pixels. The edges on the chip are separated into two polarities, namely, right-side-bright (R) and left-side-bright (L), which are kept separate throughout the chip, including the motion circuitry. For comparison, in biology, edges are often separated into light-on edges and light-off edges. The motion circuits are sensitive only to the motion of those edges from which they receive inputs. They detect the motion of a zero-crossing from one location to an adjacent location using a Reichardt scheme as shown in Figure 1b. Each motion detecting unit receives two zero-crossing inputs $ZC_n$ and $ZC_{n+2}$[1]. The ON-cells detect the onset of zero-crossings (a rising voltage edge) by firing a pulse. The units marked with D's delay these pulses by an amount D, controlled externally. The correlation units marked with X's logically AND a delayed version of a pulse from one location with an undelayed version of a pulse from the adjacent location. The output from the left correlator is sensitive to motion from location $n + 2$ to location $n$ since the motion delay is compensated by the built-in circuit delay. The outputs of the two correlators are subtracted to yield the final motion signal.          Figure 2a shows the circuit details. The boxes labelled with pulse symbols represent axon circuits. The

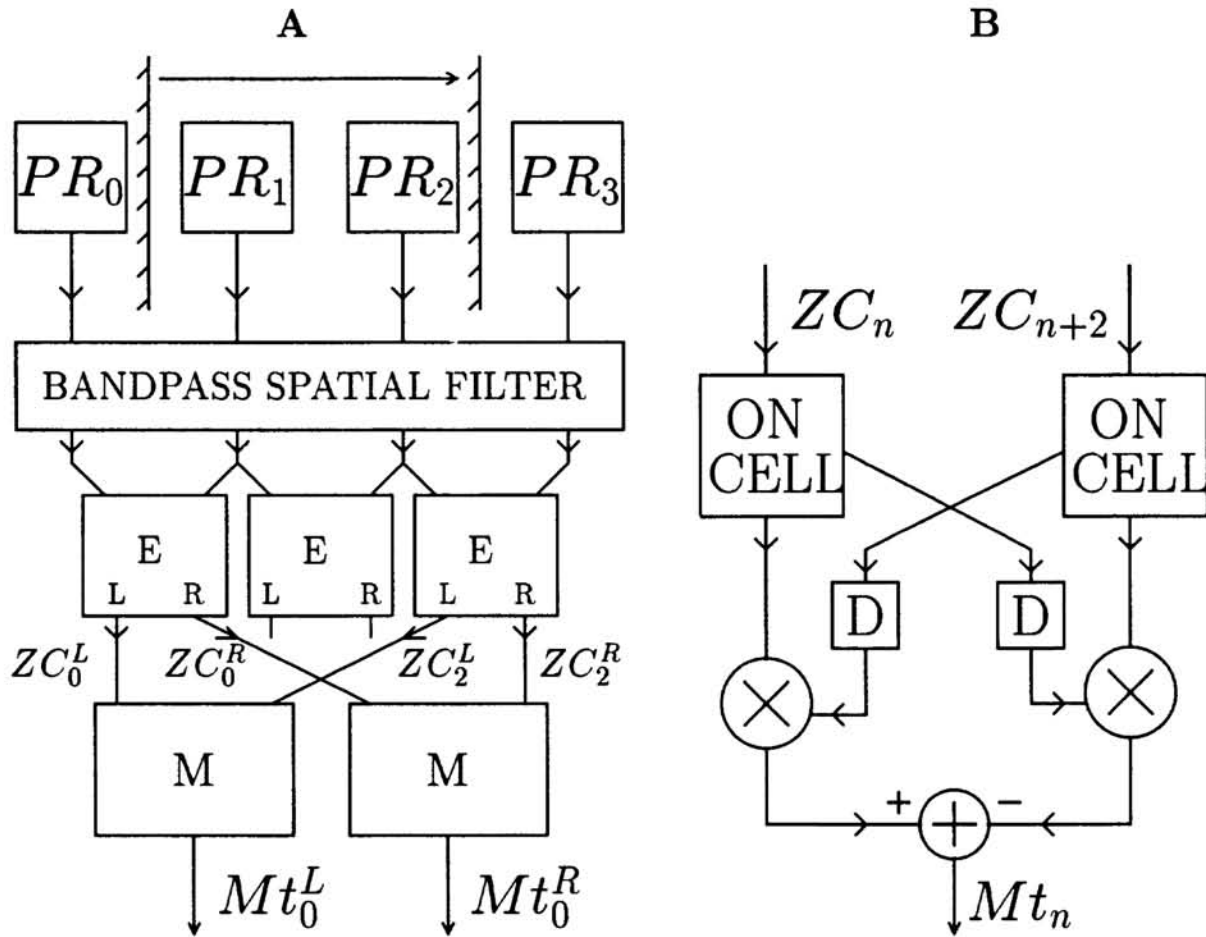

Figure 1—(A) The bandpass filtered photoreceptor signal is fed to the edge detectors marked with E's. The motion of these edges is detected by the motion detecting units marked with M's. (B) A single motion detecting unit, corresponding to a "M" unit in fig. A, has a Reichardt-like architecture.

axon circuits generate a single pulse of externally controlled width, $P$, in response to a sharp positive transition in their input, but remain inactive in response to a negative transition. In order to generate pulses that are delayed from the onset of a zero-crossing, the output of one axon circuit, with pulse width parameter $D$, is coupled via an inverter to the input of another axon circuit, with pulse width parameter $P$. The multiplication operation is implemented by a simple logical AND. The subtraction operation is implemented by a subtraction of two currents. An off-chip sense amplifier converts the bidirectional current pulse outputs of the local motion detectors into active-low or active-high voltage pulses. The axon circuit is shown in Figure 2b. Further details of its operation may be found in (Sarpeshkar et al., 1992).

Figure 3a shows how the velocity tuning curve is obtained. If the image velocity, $v$, is positive, and $\Delta x$ is the distance between adjacent zero-crossing locations, then it can be shown that the output pulse width for the positive-velocity half of the motion detector, $t_p$, is

$$t_p = u\Theta(u), \tag{1}$$

where

$$u = P - \mid \frac{\Delta x}{v} - D \mid, \tag{2}$$

and $\Theta(,)$ is the unit step function. If $v$ is negative, the same eqns. apply for the negative-velocity half of the motion detector except that the signs of $\Delta x$ and $t_p$ are reversed.

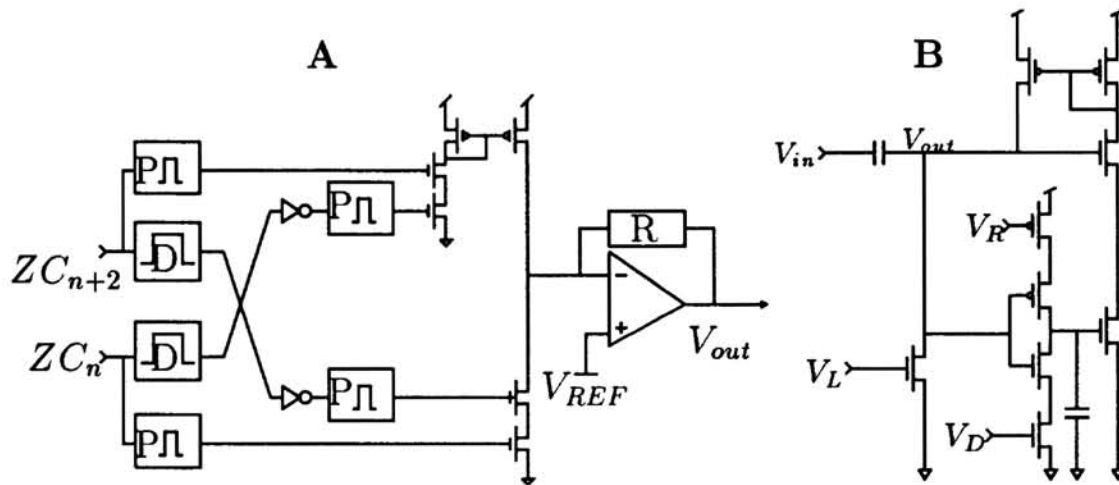

Figure 2—(A) The circuitry implementing the Reichardt scheme of Figure 1b, is shown. The boxes labelled $P$ and $D$ represent axon circuits of pulse width parameter $P$ and $D$, respectively. (B) The circuit details of an axon circuit that implements an ON-cell of Figure 1b are shown. The input and output are $V_{in}$ and $V_{out}$ respectively. The circuit was designed to mimic the behavior of sodium and leak conductances at the node of Ranvier in an axon fiber. The pulse width of the output pulse, the refractory period following its generation, and the threshold height of the input edge needed to trigger the pulse are determined by the values of bias voltages $V_D$, $V_R$ and $V_L$ respectively.

## Experimental Data

Figure 4a shows the outputs of motion detectors between zero-crossings 3 and 5, 7 and 9, and 11 and 13, denoted as $Mt_3$, $Mt_7$, and $Mt_{11}$, respectively. For an edge passing from left to right, the outputs $Mt_{11}$, $Mt_7$ and $Mt_3$ are excited in this order, and they each report a positive velocity (active high output that is above $V_{REF}$). For an edge passing from right to left, the outputs $Mt_3$, $Mt_7$ and $Mt_{11}$ are excited in this order and they each report a negative velocity (active low output that is below $V_{REF}$). Note that the amplitudes of these pulses contain no speed information and only signal the direction of motion. Figure 4b shows that the output $Mt_3$ is tuned to a particular velocity. As the rotational frequency of a cylinder with a painted edge is decreased from a velocity corresponding to a motor voltage of - 6.1V to a velocity corresponding to a motor voltage of -1.3V, the output pulse width increases, then decreases again, as the optimal velocity is traversed through. A similar tuning curve is observed for positive motor voltages.    If the distance from the surface of the spinning cylinder to the center of the lens is $o$, the distance from the center of the lens to the chip is $i$, the radius of the spinning cylinder is $R$, and its frequency

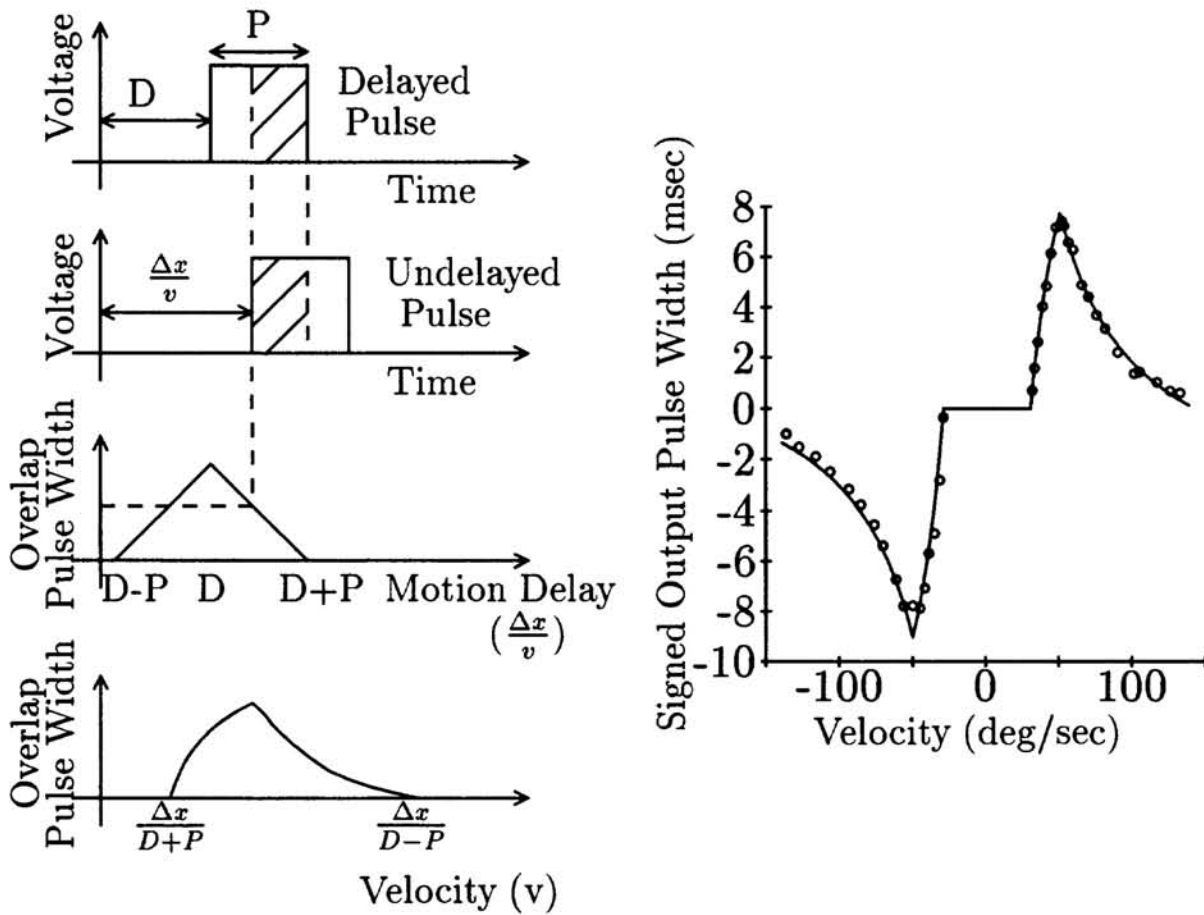

**Figure 3**—(a) The figure shows how the overlap between the delayed and unde-layed pulses gives rise to velocity tuning for motion in the preferred direction. (b) The figure shows experimental data (circles) and a theoretical fit (line) for the motion unit $Mt_3$'s output response vs. angular velocity.

of rotation is $f$, then the velocity, $v$, of the moving edge as seen by the chip is given by

$$v = 2\pi f R \frac{i}{o}. \tag{3}$$

The angular velocity, $\omega$, of the moving edge, is

$$\omega = \frac{2\pi f R}{o} = \frac{v}{\frac{o}{i}}. \tag{4}$$

Figure 3b shows the output pulse width of $Mt_3$ plotted against the angular velocity of the edge ($\omega$). The data are fit by a curve that computes $t_p$ vs. $\omega$ from (1)-(4), using measured values of $\Delta x = 180 \ \mu m$, $o = 310$ mm, $i = 17$ mm, and $R = 58$ mm.

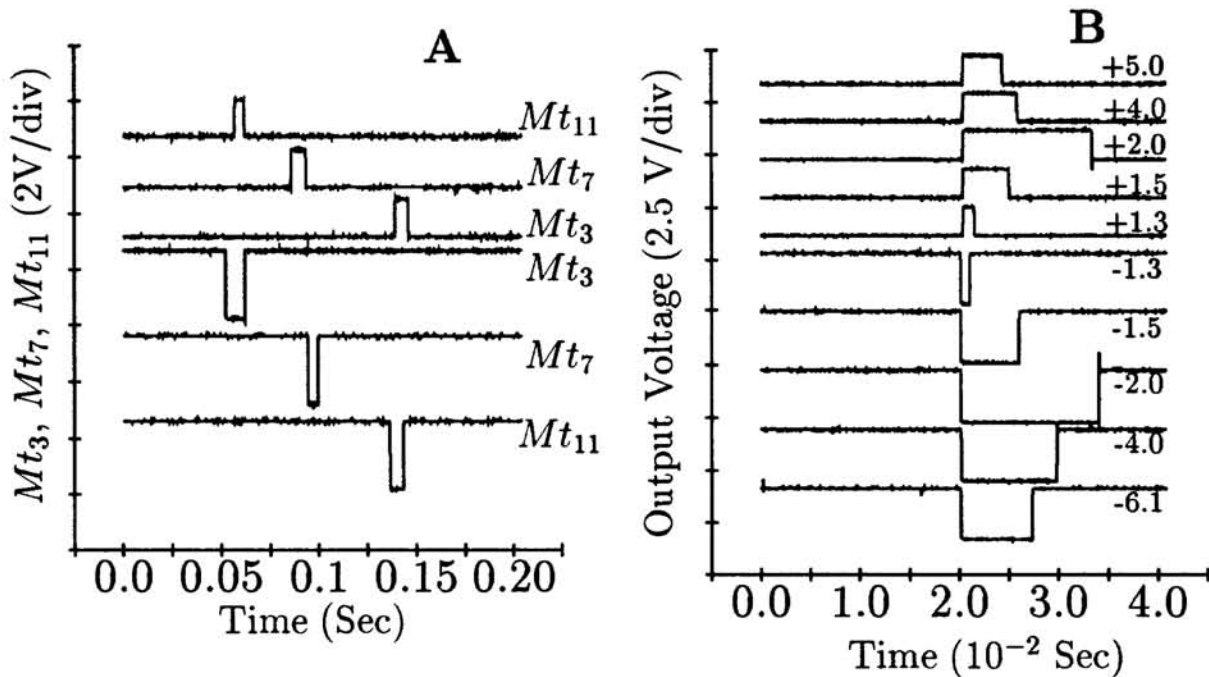

**Figure 4**—(A) The chip output waveforms are active-high when the motion is in a direction such that motion detectors 11, 7 and 3 are stimulated in that order. In the opposite direction (3 → 7 → 11), the output waveforms are active-low. (B) The figure demonstrates the tuned velocity behavior of the motion unit $Mt_3$ for various motor voltages (in V). Large positive voltages correspond to fast motion in one direction and large negative voltages correspond to fast motion in the opposite direction.

## 3    The Facilitate-and-Trigger Scheme

Although the delay-and-correlate scheme works well, the output yields ambiguous motion information due to its non-monotonic velocity dependence, i.e., we don't know if the output is small because the velocity is too slow or because it is too fast. This problem may be solved by aggregating the outputs of a series of motion detectors with overlapping tuning curves and progressively larger optimal velocities. This solution is plausible in physiology but expensive in VLSI. We were thus motivated to create a new motion detector that possessed a monotonic velocity tuning curve from the outset.

Figure 5a shows the architecture of such a motion detecting unit: The need for computing spatial edges is obviated by having a photoreceptor sensitive to temporal features caused by moving edges, i.e., sharp light onsets/offsets (Delbrück, 1993). Each ON-cell fires a pulse in response to the onset of a temporal edge. The pulses are fed to the facilitatory (F) and trigger (T) inputs of motion detectors tuned for motion in the left or right directions. The facilitatory pulse defines a time window of externally controlled width $F$, within which the output motion pulse may be activated by the trigger pulse. Thus, the rising edge of the trigger pulse must occur within the time window set by the facilitatory pulse in order to create a motion

output. If this condition is satisfied, the output motion pulse is triggered (begins) at the start of the trigger pulse and ends at the end of the facilitatory pulse; its width, thus, encodes the arrival time difference between the onset pulses at adjacent locations due to the motion delay. Each half of the motion detector only responds to motion in the direction that corresponds to *F before T*. Figure 5c shows that the velocity tuning in this scheme is monotonic. It can be shown that the output pulse width for the positive half of the motion detector, $t_p$, is given by

$$t_p = u\Theta(u),\qquad(5)$$

where $u$ is given by,

$$u = F - \frac{\Delta x}{v}.\qquad(6)$$

The dead zone near the origin may be made as small as needed by increasing the width of the facilitatory pulse.          Figure 5b shows a compact circuit that

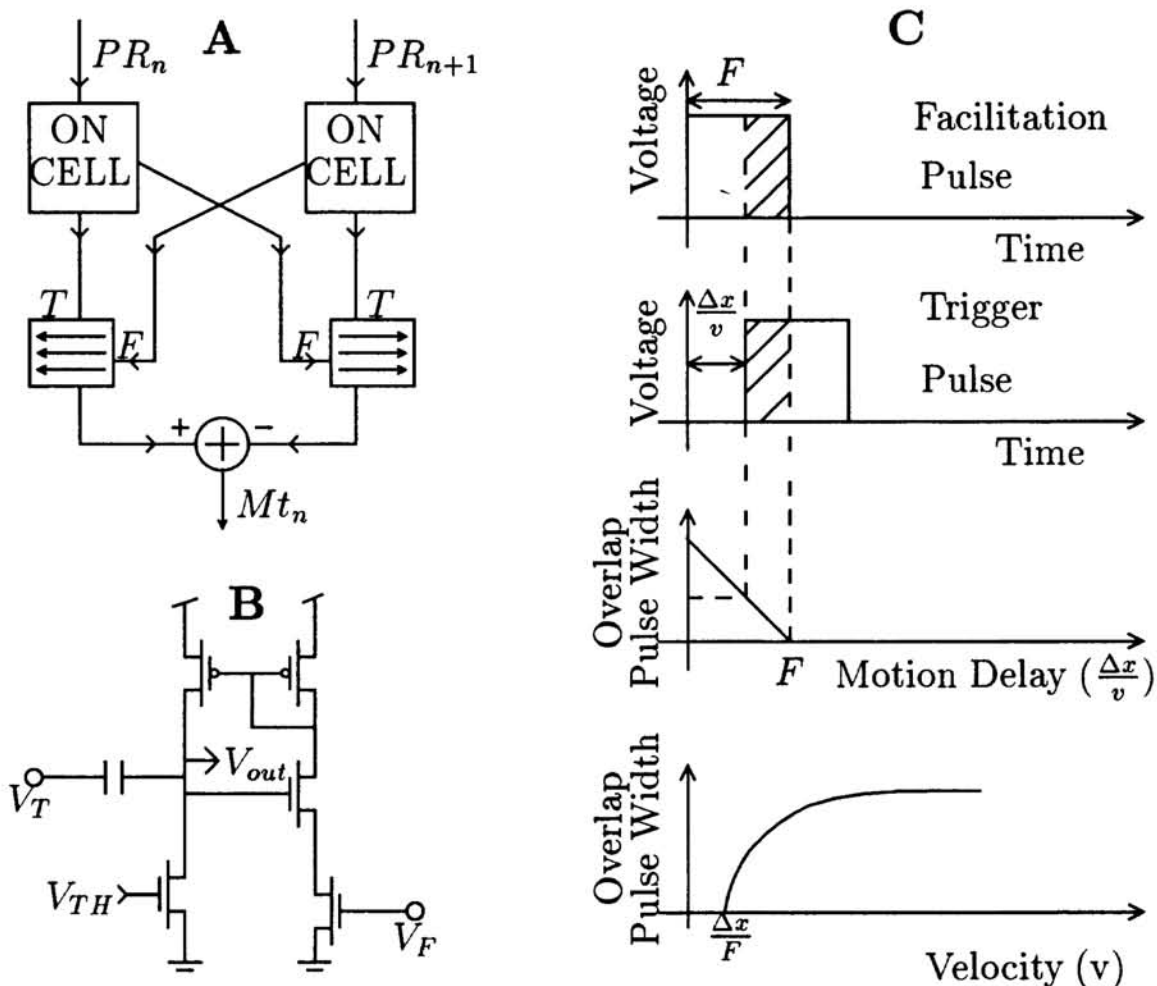

**Figure 5**—(A) **The motion detecting unit uses a facilitate-and-trigger paradigm instead of a delay-and-correlate paradigm as a basis for its operation. (B) A compact circuit implements the motion detecting unit.** $V_T$ **is the trigger input,** $V_F$ **is the facilitatory input,** $V_{out}$ **is the output and** $V_{TH}$ **is a bias voltage. (C) The output response has a monotonic dependence on velocity.**

implements the facilitate-and-trigger scheme. The ON-cell, subtraction and sense-amplifier circuitry are implemented as in the Reichardt scheme.

The facilitate-and-trigger approach requires a total of 32 transistors per motion unit compared with a total of 64 in the delay-and-correlate approach, needs one parameter to be controlled ($F$) rather than two ($D$ and $P$), and yields monotonic tuning from the outset. We, therefore, believe that it will prove to be the superior of the two approaches.

## 4    Conclusions

The evaluations of onsets, delays and coincidences, required for computing motion are implemented very naturally with pulses rather than by graded potentials as in all other motion chips, built so far. Both of our motion algorithms time the motion of image features in an efficient fashion by using pulse computation.

### Acknowledgements

Many thanks to Carver Mead for his encouragement, support and use of lab facilities. We acknowledge useful discussions with William Bialek, Nicola Franceschini and Tobias Delbrück. This work was supported by grants from the Office of Naval Research and the California Competitive Technologies Program. Chip fabrication was provided by MOSIS.

## Footnotes

[1] $ZC_{n+1}$ could have been used as well. $ZC_{n+2}$ was chosen due to wiring constraints and because it increases the baseline distance for computing motion.

### References

W. Bair, and C. Koch, "An Analog VLSI Chip for Finding Edges from Zero-crossings.", In *Advances in Neural Information Processing Systems Vol. 3*, R. Lippman, J. Moody, D. Touretzky, eds., pp. 399-405, Morgan Kaufmann, San Mateo, CA, 1991.

T. Delbrück, "Investigations of Analog VLSI Visual Transduction and Motion Processing", PhD. thesis, Computation and Neural Systems Program, Caltech, Pasadena, CA, 1993.

B. Hassenstein and W. Reichardt, "Systemtheoretische Analyse der Zeit, Reihenfolgen, und Vorzeichenauswertung bei der Bewegungsperzepion des Rüsselkäfers *Chlorophanus*", *Z. Naturforsch.*11b: pp. 513-524, 1956.

T. Horiuchi, W. Bair, A. Moore, B. Bishofberger, J. Lazzaro, C. Koch, "Computing Motion Using Analog VLSI Vision Chips: an Experimental Comparison among Different Approaches", Intl. Journal of Computer Vision, **8**, pp. 203-216, 1992.

D. Marr and S. Ullman, "Directional Selectivity and its Use in Early Visual Processing", *Proc. R. Soc. Lond B* **211**, pp. 151-180, 1981.

R. Sarpeshkar, L. Watts, C. Mead, "Refractory Neuron Circuits", Internal Lab Memorandum, Physics of Computation Laboratory, Pasadena, CA, 1992.

J. Tanner and C. Mead, "An Integrated Optical Motion Sensor", *VLSI Signal Processing II*, S-Y Kung, R.E. Owen, and J.G. Nash, eds., 59-76, IEEE Press, NY, 1986.